# Self-calibrating Probability Forecasting

**Vladimir Vovk**
Computer Learning Research Centre
Department of Computer Science
Royal Holloway, University of London
Egham, Surrey TW20 0EX, UK
vovk@cs.rhul.ac.uk

**Glenn Shafer**
Rutgers School of Business
Newark and New Brunswick
180 University Avenue
Newark, NJ 07102, USA
gshafer@andromeda.rutgers.edu

**Ilia Nouretdinov**

Computer Learning Research Centre
Department of Computer Science
Royal Holloway, University of London
Egham, Surrey TW20 0EX, UK
ilia@cs.rhul.ac.uk

## Abstract

In the problem of probability forecasting the learner's goal is to output, given a training set and a new object, a suitable probability measure on the possible values of the new object's label. An on-line algorithm for probability forecasting is said to be well-calibrated if the probabilities it outputs agree with the observed frequencies. We give a natural non-asymptotic formalization of the notion of well-calibratedness, which we then study under the assumption of randomness (the object/label pairs are independent and identically distributed). It turns out that, although no probability forecasting algorithm is automatically well-calibrated in our sense, there exists a wide class of algorithms for "multiprobability forecasting" (such algorithms are allowed to output a set, ideally very narrow, of probability measures) which satisfy this property; we call the algorithms in this class "Venn probability machines". Our experimental results demonstrate that a 1-Nearest Neighbor Venn probability machine performs reasonably well on a standard benchmark data set, and one of our theoretical results asserts that a simple Venn probability machine asymptotically approaches the true conditional probabilities regardless, and without knowledge, of the true probability measure generating the examples.

## 1 Introduction

We are interested in the on-line version of the problem of probability forecasting: we observe pairs of objects and labels sequentially, and after observing the $n$th object $x_n$ the goal is to give a probability measure $p_n$ for its label; as soon as $p_n$ is output, the label $y_n$ of $x_n$ is disclosed and can be used for computing future probability forecasts. A good

review of early work in this area is Dawid [1]. In this introductory section we will assume that $y_n \in \{0, 1\}$; we can then take $p_n$ to be a real number from the interval $[0, 1]$ (the probability that $y_n = 1$ given $x_n$); our exposition here will be very informal.

The standard view ( [1], pp. 213–216) is that the quality of probability forecasting systems has two components: "reliability" and "resolution". At the crudest level, reliability requires that the forecasting system should not lie, and resolution requires that it should say something useful. To be slightly more precise, consider the first $n$ forecasts $p_i$ and the actual labels $y_i$.

The most basic test is to compare the overall average forecast probability $\bar{p}_n :=$ $n^{-1} \sum_{i=1}^{n} p_i$ with the overall relative frequency $\bar{y}_n := n^{-1} \sum_{i=1}^{n} y_i$ of 1s among $y_i$. If $\bar{p}_n \approx \bar{y}_n$, the forecasts are "unbiased in the large".

A more refined test would look at the subset of $i$ for which $p_i$ is close to a given value $p^*$, and compare the relative frequency of $y_i = 1$ in this subset, say $\bar{y}_n(p^*)$, with $p^*$. If

$$\bar{y}_n(p^*) \approx p^* \text{ for all } p^*, \tag{1}$$

the forecasts are "unbiased in the small", "reliable", "valid", or "well-calibrated"; in later sections, we will use "well-calibrated", or just "calibrated", as a technical term. Forecasting systems that pass this test at least get the frequencies right; in this sense they do not lie.

It is easy to see that there are reliable forecasting systems that are virtually useless. For example, the definition of reliability does not require that the forecasting system pay any attention to the objects $x_i$. In another popular example, the labels follow the pattern

$$y_i = \begin{cases} 1 & \text{if } i \text{ is odd} \\ 0 & \text{otherwise.} \end{cases}$$

The forecasts $p_i = 0.5$ are reliable, at least asymptotically (0.5 is the right relative frequency) but not as useful as $p_1 = 1, p_2 = 0, \ldots$; the "resolution" (which we do not define here) of the latter forecasts is better.

In this paper we construct forecasting systems that are automatically reliable. To achieve this, we allow our prediction algorithms to output sets of probability measures $P_n$ instead of single measures $p_n$; typically the sets $P_n$ will be small (see §5).

This paper develops the approach of [2–4], which show that it is possible to produce valid, asymptotically optimal, and practically useful p-values; the p-values can be then used for region prediction. Disadvantages of p-values, however, are that their interpretation is less direct than that of probabilities and that they are easy to confuse with probabilities; some authors have even objected to any use of p-values (see, e.g., [5]). In this paper we use the methodology developed in the previous papers to produce valid probabilities rather than p-values.

All proofs are omitted and can be found in [6].

## 2 Probability forecasting and calibration

From this section we start rigorous exposition. Let $\mathcal{P}(\mathbf{Y})$ be the set of all probability measures on a measurable space $\mathbf{Y}$. We use the following protocol in this paper:

MULTIPROBABILITY FORECASTING

**Players:** Reality, Forecaster

**Protocol:**

FOR $n = 1, 2, \ldots$:
    Reality announces $x_n \in \mathbf{X}$.
    Forecaster announces $P_n \subseteq \mathcal{P}(\mathbf{Y})$.
    Reality announces $y_n \in \mathbf{Y}$.

In this protocol, Reality generates *examples* $z_n = (x_n, y_n) \in \mathbf{Z} := \mathbf{X} \times \mathbf{Y}$ consisting of two parts, *objects* $x_n$ and *labels* $y_n$. After seeing the object $x_n$ Forecaster is required to output a prediction for the label $y_n$. The usual probability forecasting protocol requires that Forecaster output a probability measure; we relax this requirement by allowing him to output a family of probability measures (and we are interested in the case where the families $P_n$ become smaller and smaller as $n$ grows). It can be shown (we omit the proof and even the precise statement) that it is impossible to achieve automatic well-calibratedness, in our finitary sense, in the probability forecasting protocol.

In this paper we make the simplifying assumption that the label space $\mathbf{Y}$ is finite; in many informal explanations it will be assumed binary, $\mathbf{Y} = \{0, 1\}$. To avoid unnecessary technicalities, we will also assume that the families $P_n$ chosen by Forecaster are finite and have no more than $K$ elements; they will be represented by a list of length $K$ (elements in the list can repeat). A *probability machine* is a measurable strategy for Forecaster in our protocol, where at each step he is required to output a sequence of $K$ probability measures.

The problem of calibration is usually treated in an asymptotic framework. Typical asymptotic results, however, do not say anything about finite data sequences; therefore, in this paper we will only be interested in the non-asymptotic notion of calibration. All needed formal definitions will be given, but space limitations prevent us from including detailed explanations and examples, which can be found in [6].

Let us first limit the duration of the game, replacing $n = 1, 2, \ldots$ in the multiprobability forecasting protocol by $n = 1, \ldots, N$ for a finite horizon $N$. It is clear that, regardless of formalization, we cannot guarantee that miscalibration, in the sense of (1) being violated, will never happen: for typical probability measures, everything can happen, perhaps with a small probability. The idea of our definition is: a prediction algorithm is well-calibrated if any evidence of miscalibration translates into evidence against the assumption of randomness. Therefore, we first need to define ways of testing calibration and randomness; this will be done following [7].

A *game N-martingale* is a function $M$ on sequences of the form $x_1, p_1, y_1, \ldots, x_n, p_n, y_n$, where $n = 0, \ldots, N$, $x_i \in \mathbf{X}$, $p_i \in \mathcal{P}(\mathbf{Y})$, and $y_i \in \mathbf{Y}$, that satisfies

$$M(x_1, p_1, y_1, \ldots, x_{n-1}, p_{n-1}, y_{n-1}) = \int_{\mathbf{Y}} M(x_1, p_1, y_1, \ldots, x_n, p_n, y) p_n(dy)$$

for all $x_1, p_1, y_1, \ldots, x_n, p_n$, $n = 1, \ldots, N$. A *calibration N-martingale* is a nonnegative game $N$-martingale that is invariant under permutations:

$$M(x_1, p_1, y_1, \ldots, x_N, p_N, y_N) = M(x_{\pi(1)}, p_{\pi(1)}, y_{\pi(1)}, \ldots, x_{\pi(N)}, p_{\pi(N)}, y_{\pi(N)})$$

for any $x_1, p_1, y_1, \ldots, x_N, p_N, y_N$ and any permutation $\pi : \{1, \ldots, N\} \to \{1, \ldots, N\}$.

To cover the multiprobability forecasting protocol, we extend the domain of definition for a calibration $N$-martingale $M$ from sequences of the form $x_1, p_1, y_1, \ldots, x_n, p_n, y_n$, where $p_1, \ldots, p_n$ are single probability measures on $\mathbf{Y}$, to sequences of the form $x_1, P_1, y_1, \ldots, x_n, P_n, y_n$, where $P_1, \ldots, P_n$ are sets of probability measures on $\mathbf{Y}$, by

$$M(x_1, P_1, y_1, \ldots, x_n, P_n, y_n) := \inf_{p_1 \in P_1, \ldots, p_n \in P_n} M(x_1, p_1, y_1, \ldots, x_n, p_n, y_n).$$

A $Q^N$-*martingale*, where $Q$ is a probability measure on $\mathbf{Z}$, is a function $S$ on sequences of the form $x_1, y_1, \ldots, x_n, y_n$, where $n = 0, \ldots, N$, $x_i \in \mathbf{X}$, and $y_i \in \mathbf{Y}$, that satisfies

$$S(x_1, y_1, \ldots, x_{n-1}, y_{n-1}) = \int_{\mathbf{Z}} S(x_1, y_1, \ldots, x_{n-1}, y_{n-1}, x, y) Q(dx, dy)$$

for all $x_1, y_1, \ldots, x_{n-1}, y_{n-1}$, $n = 1, \ldots, N$.

If a nonnegative $Q^N$-martingale $S$ starts with $S(\square) = 1$ and ends with $S(x_1, y_1, \ldots, y_N)$ very large, then we may reject $Q$ as the probability measure generating individual examples $(x_n, y_n)$. This interpretation is supported by Doob's inequality. Analogously, if a game $N$-martingale $M$ starts with $M(\square) = 1$ and ends with $M(x_1, P_1, y_1, \ldots, y_N)$ very large, then we may reject the hypothesis that each $P_n$ contains the true probability measure for $y_n$. If $M$ is a calibration $N$-martingale, this event is interpreted as evidence of miscalibration. (The restriction to calibration $N$-martingales is motivated by the fact that (1) is invariant under permutations).

We call a probability machine $F$ $N$-*calibrated* if for any probability measure $Q$ on $\mathbf{Z}$ and any nonnegative calibration $N$-martingale $M$ with $M(\square) = 1$, there exists a $Q^N$-martingale $S$ with $S(\square) = 1$ such that

$$M(x_1, F(x_1), y_1, \ldots, x_N, F(x_1, y_1, \ldots, x_N), y_N) \le S(x_1, y_1, \ldots, x_N, y_N)$$

for all $x_1, y_1, \ldots, x_N, y_N$. We say that $F$ is *finitarily calibrated* if it is $N$-calibrated for each $N$.

## 3 Self-calibrating probability forecasting

Now we will describe a general algorithm for multiprobability forecasting. Let $\mathbb{N}$ be the sets of all positive integer numbers. A sequence of measurable functions $A_n : \mathbf{Z}^n \to \mathbb{N}^n$, $n = 1, 2, \ldots$, is called a *taxonomy* if, for any $n \in \mathbb{N}$, any permutation $\pi$ of $\{1, \ldots, n\}$, any $(z_1, \ldots, z_n) \in \mathbf{Z}^n$, and any $(\alpha_1, \ldots, \alpha_n) \in \mathbb{N}^n$,

$$(\alpha_1, \ldots, \alpha_n) = A_n(z_1, \ldots, z_n) \implies (\alpha_{\pi(1)}, \ldots, \alpha_{\pi(n)}) = A_n(z_{\pi(1)}, \ldots, z_{\pi(n)}).$$

In other words,

$$A_n : (z_1, \ldots, z_n) \mapsto (\alpha_1, \ldots, \alpha_n) \tag{2}$$

is a taxonomy if every $\alpha_i$ is determined by the bag[1] $\lfloor z_1, \ldots, z_n \rceil$ and $z_i$. We let $|B|$ stand for the number of elements in a set $B$. The *Venn probability machine associated with* $(A_n)$ is the probability machine which outputs the following $K = |\mathbf{Y}|$ probability measures $p_y$, $y \in \mathbf{Y}$, at the $n$th step: complement the new object $x_n$ by the postulated label $y$; consider the division of $\lfloor z_1, \ldots, z_n \rceil$, where $z_n$ is understood (only for the purpose of this definition) to be $(x_n, y)$, into groups (formally, bags) according to the values of $A_n$ (i.e., $z_i$ and $z_j$ are assigned to the same group if and only if $\alpha_i = \alpha_j$, where the $\alpha$s are defined by (2)); find the empirical distribution $p_y \in \mathcal{P}(\mathbf{Y})$ of the labels in the group $G$ containing the $n$th example $z_n = (x_n, y)$:

$$p_y(\{y'\}) := \frac{|\{(x^*, y^*) \in G : y^* = y'\}|}{|G|}.$$

A *Venn probability machine* (VPM) is the Venn probability machine associated with some taxonomy.

**Theorem 1** *Any Venn probability machine is finitarily calibrated.*

It is clear that VPM depends on the taxonomy only through the way it splits the examples $z_1, \ldots, z_n$ into groups; therefore, we may specify only the latter when constructing specific VPMs.

**Remark** The notion of VPM is a version of Transductive Confidence Machine (TCM) introduced in [8] and [9], and Theorem 1 is a version of Theorem 1 in [2].

## 4 Discussion of the Venn probability machine

In this somewhat informal section we will discuss the intuitions behind VPM, considering only the binary case $\mathbf{Y} = \{0, 1\}$ and considering the probability forecasts $p_i$ to be elements of $[0, 1]$ rather than $\mathcal{P}(\{0, 1\})$, as in §1. We start with the almost trivial *Bernoulli* case, where the objects $x_i$ are absent,[2] and our goal is to predict, at each step $n = 1, 2, \ldots$, the new label $y_n$ given the previous labels $y_1, \ldots, y_{n-1}$. The most naive probability forecast is $p_n = k/(n - 1)$, where $k$ is the number of 1s among the first $n - 1$ labels. (Often "regularized" forms of $k/(n - 1)$, such as Laplace's rule of succession $(k + 1)/(n + 1)$, are used.)

In the Bernoulli case there is only one natural VPM: the multiprobability forecast for $y_n$ is $\{k/n, (k+1)/n\}$. Indeed, since there are no objects $x_n$, it is natural to take the one-element taxonomy $A_n$ at each step, and this produces the VPM $P_n = \{k/n, (k + 1)/n\}$. It is clear that the diameter $1/n$ of $P_n$ for this VPM is the smallest achievable. (By the *diameter* of a set we mean the supremum of distances between its points.)

Now let us consider the case where $x_n$ are present. The probability forecast $k/(n - 1)$ for $y_n$ will usually be too crude, since the known population $z_1, \ldots, z_{n-1}$ may be very heterogeneous. A reasonable statistical forecast would take into account only objects $x_i$ that are similar, in a suitable sense, to $x_n$. A simple modification of the Bernoulli forecast $k/(n - 1)$ is as follows:

1. Split the available objects $x_1, \ldots, x_n$ into a number of groups.
2. Output $k'/n'$ as the predicted probability that $y_n = 1$, where $n'$ is the number of objects among $x_1, \ldots, x_{n-1}$ in the same group as $x_n$ and $k'$ is the number of objects among those $n'$ that are labeled as 1.

At the first stage, a delicate balance has to be struck between two contradictory goals: the groups should be as large as possible (to have a reasonable sample size for estimating probabilities); the groups should be as homogeneous as possible. This problem is sometimes referred to as the "reference class problem"; according to Kılınç [10], John Venn was the first to formulate and analyze this problem with due philosophical depth.

The procedure offered in this paper is a simple modification of the standard procedure described in the previous paragraph:

0. Consider the two possible completions of the known data

$$(z_1, \ldots, z_{n-1}, x_n) = ((x_1, y_1), \ldots, (x_{n-1}, y_{n-1}), x_n) :$$

in one (called the 0-*completion*) $x_n$ is assigned label 0, and in the other (called the 1-*completion*) $x_n$ is assigned label 1.

1. In each completion, split all examples $z_1, \ldots, z_{n-1}, (x_n, y)$ into a number of groups, so that the split does not depend on the order of examples ($y = 0$ for the 0-partition and $y = 1$ for the 1-partition).

2. In each completion, output $k'/n'$ as the predicted probability that $y_n = 1$, where $n'$ is the number of examples among $z_1, \ldots, z_{n-1}, (x_n, y)$ in the same group as $(x_n, y)$ and $k'$ is the number of examples among those $n'$ that are labeled as $1$.

In this way, we will have not one but two predicted probabilities that $y_n = 1$; but in practically interesting cases we can hope that these probabilities will be close to each other (see the next section).

Venn's reference class problem reappears in our procedure as the problem of avoiding over- and underfitting. A taxonomy with too many groups means overfitting; it is punished by the large diameter of the multiprobability forecast (importantly, this is visible, unlike the standard approaches). Too few groups means underfitting (and poor resolution).

Important advantages of our procedure over the naive procedure are: our procedure is self-calibrating; there exists an asymptotically optimal VPM (see §6); we can use labels in splitting examples into groups (this will be used in the next section).

## 5  Experiments

In this section, we will report the results for a natural taxonomy applied to the well-known USPS data set of hand-written digits; this taxonomy is inspired by the 1-Nearest Neighbor algorithm. First we describe the taxonomy, and then the way in which we report the results for the VPM associated with this taxonomy.

Since the data set is relatively small (9298 examples in total), we have to use a crude taxonomy: two examples are assigned to the same group if their nearest neighbors have the same label; therefore, the taxonomy consists of 10 groups. The distance between two examples is defined as the Euclidean distance between their objects (which are $16 \times 16$ matrices of pixels and represented as points in $\mathbb{R}^{256}$).

The algorithm processes the $n$th object $x_n$ as follows. First it creates the $10 \times 10$ matrix $A$ whose entry $A_{i,j}$, $i, j = 0, \ldots, 9$, is computed by assigning $i$ to $x_n$ as label and finding the fraction of examples labeled $j$ among the examples in the bag $\{z_1, \ldots, z_{n-1}, (x_n, i)\}$ belonging to the same group as $(x_n, i)$. The *quality* of a column of this matrix is its minimum entry. Choose a column (called the *best* column) with the highest quality; let the best column be $j_{\text{best}}$. Output $j_{\text{best}}$ as the prediction and output

$$\left[ \min_{i=0,\ldots,9} A_{i,j_{\text{best}}}, \max_{i=0,\ldots,9} A_{i,j_{\text{best}}} \right]$$

as the interval for the probability that this prediction is correct. If the latter interval is $[a, b]$, the complementary interval $[1-b, 1-a]$ is called the *error probability interval*. In Figure 1 we show the following three curves: the cumulative error curve $E_n := \sum_{i=1}^{n} \text{err}_i$, where $\text{err}_i = 1$ if an error (in the sense $j_{\text{best}} \neq y_i$) is made at step $i$ and $\text{err}_i = 0$ otherwise; the *cumulative lower error probability curve* $L_n := \sum_{i=1}^{n} l_i$ and the *cumulative upper error probability curve* $U_n := \sum_{i=1}^{n} u_i$, where $[l_i, u_i]$ is the error probability interval output by the algorithm for the label $y_i$. The values $E_n$, $L_n$ and $U_n$ are plotted against $n$. The plot confirms that the error probability intervals are calibrated.

## 6  Universal Venn probability machine

The following result asserts the existence of a universal VPM. Such a VPM can be constructed quite easily using the histogram approach to probability estimation [11].

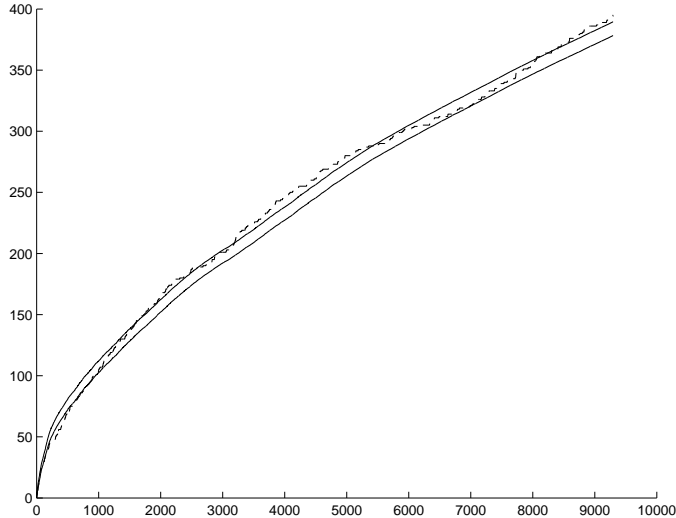

Figure 1: On-line performance of the 1-Nearest Neighbor VPM on the USPS data set (9298 hand-written digits, randomly permuted). The dashed line shows the cumulative number of errors $E_n$ and the solid ones the cumulative upper and lower error probability curves $U_n$ and $L_n$. The mean error $E_N/N$ is 0.0425 and the mean probability interval $(1/N)[L_N, U_N]$ is $[0.0407, 0.0419]$, where $N = 9298$ is the size of the data set. This figure is not significantly affected by statistical variation (due to the random choice of the permutation of the data set).

**Theorem 2** *Suppose* **X** *is a Borel space. There exists a VPM such that, if the examples are generated from* $Q^\infty$,

$$\sup_{p \in P_n} \rho(Q(\cdot \mid x_n), p) \to 0 \quad (n \to \infty)$$

*in probability, where* $\rho$ *is the variation distance,* $Q(\cdot \mid x_n)$ *is a fixed version of the regular conditional probabilities for* $y_n$ *given* $x_n$, *and* $P_n$ *are the multiprobabilities produced by the VPM.*

This theorem shows that not only all VPMs are reliable but some of them also have asymptotically optimal resolution. The version of this result for p-values was proved in [4].

## 7 Comparisons

In this section we briefly and informally compare this paper's approach to standard approaches in machine learning.

Two most important approaches to analysis of machine-learning algorithms are Bayesian learning theory and PAC theory (the recent mixture, the PAC-Bayesian theory, is part of PAC theory in its assumptions). This paper is in a way intermediate between Bayesian learning (no empirical justification for probabilities is required) and PAC learning (the goal is to find or bound the true probability of error, not just to output calibrated probabilities). An important difference of our approach from the PAC approach is that we are interested in the conditional probabilities for the label given the new object, whereas PAC theory (even in its "data-dependent" version, as in [12–14]) tries to estimate the unconditional probability of error.

## Acknowledgments

We are grateful to Phil Dawid for a useful discussion and to the anonymous referees for suggestions for improvement. This work was partially supported by EPSRC (grant GR/R46670/01), BBSRC (grant 111/BIO14428), and EU (grant IST-1999-10226).

## Footnotes

[1] By "bag" we mean a collection of elements, not necessarily distinct. "Bag" and "multiset" are synonymous, but we prefer the former term in order not to overload the prefix "multi".

[2] Formally, this correspond in our protocol to the situation where $|\mathbf{X}| = 1$, and so $x_n$, although nominally present, do not carry any information.

## References

[1] A. Philip Dawid. Probability forecasting. In Samuel Kotz, Norman L. Johnson, and Campbell B. Read, editors, *Encyclopedia of Statistical Sciences*, volume 7, pages 210–218. Wiley, New York, 1986.

[2] Vladimir Vovk. On-line Confidence Machines are well-calibrated. In *Proceedings of the Forty Third Annual Symposium on Foundations of Computer Science*, pages 187–196, Los Alamitos, CA, 2002. IEEE Computer Society.

[3] Vladimir Vovk, Ilia Nouretdinov, and Alex Gammerman. Testing exchangeability on-line. In Tom Fawcett and Nina Mishra, editors, *Proceedings of the Twentieth International Conference on Machine Learning*, pages 768–775, Menlo Park, CA, 2003. AAAI Press.

[4] Vladimir Vovk. Universal well-calibrated algorithm for on-line classification. In Bernhard Schölkopf and Manfred K. Warmuth, editors, *Learning Theory and Kernel Machines: Sixteenth Annual Conference on Learning Theory and Seventh Kernel Workshop*, volume 2777 of *Lecture Notes in Artificial Intelligence*, pages 358–372, Berlin, 2003. Springer.

[5] James O. Berger and Mohan Delampady. Testing precise hypotheses (with discussion). *Statistical Science*, 2:317–352, 1987.

[6] Vladimir Vovk, Alex Gammerman, and Glenn Shafer. *Algorithmic Learning in a Random World*. Springer, New York, to appear.

[7] Glenn Shafer and Vladimir Vovk. *Probability and Finance: It's Only a Game!* Wiley, New York, 2001.

[8] Craig Saunders, Alex Gammerman, and Vladimir Vovk. Transduction with confidence and credibility. In *Proceedings of the Sixteenth International Joint Conference on Artificial Intelligence*, pages 722–726, 1999.

[9] Vladimir Vovk, Alex Gammerman, and Craig Saunders. Machine-learning applications of algorithmic randomness. In *Proceedings of the Sixteenth International Conference on Machine Learning*, pages 444–453, San Francisco, CA, 1999. Morgan Kaufmann.

[10] Berna E. Kılınç. The reception of John Venn's philosophy of probability. In Vincent F. Hendricks, Stig Andur Pedersen, and Klaus Frovin Jørgensen, editors, *Probability Theory: Philosophy, Recent History and Relations to Science*, pages 97–121. Kluwer, Dordrecht, 2001.

[11] Luc Devroye, László Györfi, and Gábor Lugosi. *A Probabilistic Theory of Pattern Recognition*. Springer, New York, 1996.

[12] Nick Littlestone and Manfred K. Warmuth. Relating data compression and learnability. Technical report, University of California, Santa Cruz, 1986.

[13] John Shawe-Taylor, Peter L. Bartlett, Robert C. Williamson, and Martin Anthony. Structural risk minimization over data-dependent hierarchies. *IEEE Transactions on Information Theory*, 44:1926–1940, 1998.

[14] David A. McAllester. Some PAC-Bayesian theorems. In *Proceedings of the Eleventh Annual Conference on Computational Learning Theory*, pages 230–234, New York, 1998. Association for Computing Machinery.
